# A Functional Architecture for Motion Pattern Processing in MSTd

**Scott A. Beardsley**
Dept. of Biomedical Engineering
Boston University
Boston, MA 02215
*sbeardsl@bu.edu*

**Lucia M. Vaina**
Dept. of Biomedical Engineering
Boston University
Boston, MA 02215
*vaina@bu.edu*

## Abstract

Psychophysical studies suggest the existence of specialized detectors for component motion patterns (radial, circular, and spiral), that are consistent with the visual motion properties of cells in the dorsal medial superior temporal area (MSTd) of non-human primates. Here we use a biologically constrained model of visual motion processing in MSTd, in conjunction with psychophysical performance on two motion pattern tasks, to elucidate the computational mechanisms associated with the processing of wide-field motion patterns encountered during self-motion. In both tasks discrimination thresholds varied significantly with the type of motion pattern presented, suggesting perceptual correlates to the preferred motion bias reported in MSTd. Through the model we demonstrate that while independently responding motion pattern units are capable of encoding information relevant to the visual motion tasks, equivalent psychophysical performance can only be achieved using interconnected neural populations that systematically inhibit non-responsive units. These results suggest the cyclic trends in psychophysical performance may be mediated, in part, by recurrent connections within motion pattern responsive areas whose structure is a function of the similarity in preferred motion patterns and receptive field locations between units.

## 1 Introduction

A major challenge in computational neuroscience is to elucidate the architecture of the cortical circuits for sensory processing and their effective role in mediating behavior. In the visual motion system, biologically constrained models are playing an increasingly important role in this endeavor by providing an explanatory substrate linking perceptual performance and the visual properties of single cells.

Single cell studies indicate the presence of complex interconnected structures in middle temporal and primary visual cortex whose most basic horizontal connections can impart considerable computational power to the underlying neural population [1, 2]. Combined psychophysical and computational studies support these findings

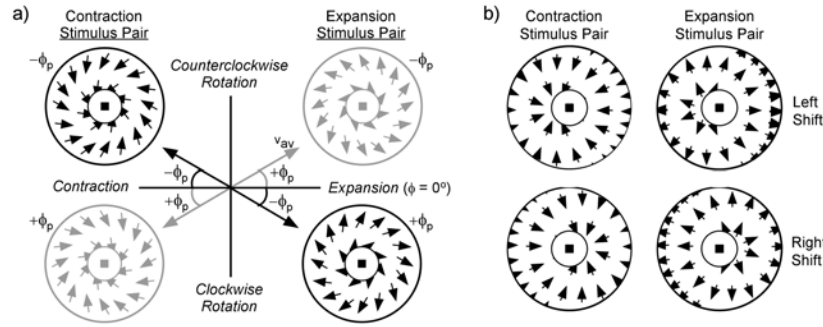

Figure 1: a) Schematic of the graded motion pattern (GMP) task. Discrimination pairs of stimuli were created by perturbing the flow angle ($\phi$) of each 'test' motion (with average dot speed, $v_{av}$), by $\pm\phi_p$ in the stimulus space spanned by radial and circular motions. b) Schematic of the shifted center-of-motion (COM) task. Discrimination pairs of stimuli were created by shifting the COM of the 'test' motion to the left and right of a central fixation point. For each motion pattern the COM was shifted within the illusory inner aperture and was never explicitly visible.

and suggest that recurrent connections may play a significant role in encoding the visual motion properties associated with various psychophysical tasks [3, 4].

Using this methodology our goal is to elucidate the computational mechanisms associated with the processing of wide-field motion patterns encountered during self-motion. In the human visual motion system, psychophysical studies suggest the existence of specialized detectors for the motion pattern components (i.e., radial, circular and spiral motions) associated with self-motion [5, 6]. Neurophysiological studies reporting neurons sensitive to motion patterns in the dorsal medial superior temporal area (MSTd) support the existence of such mechanisms [7-10], and in conjunction with psychophysical studies suggest a strong link between the patterns of neural activity and motion-based perceptual performance [11, 12].

Through the combination of human psychophysical performance and biologically constrained modeling we investigate the computational role of simple recurrent connections within a population of MSTd-like units. Based on the known visual motion properties within MSTd we ask what neural structures are computationally sufficient to encode psychophysical performance on a series of motion pattern tasks.

## 2   Motion pattern discrimination

Using motion pattern stimuli consistent with previous studies [5, 6], we have developed a set of novel psychophysical tasks designed to facilitate a more direct comparison between human perceptual performance and the visual motion properties of cells in MSTd that have been found to underlie the discrimination of motion patterns [11, 12]. The psychophysical tasks, referred to as the graded motion pattern (GMP) and shifted center-of-motion (COM) tasks, are outlined in Fig. 1.

Using a temporal two-alternative-forced-choice task we measured discrimination thresholds to global changes in the patterns of complex motion (GMP task), [13], and shifts in the center-of-motion (COM task). Stimuli were presented with central fixation using a constant stimulus paradigm and consisted of dynamic random dot displays presented in a 24$^{o}$ annular region (central 4$^{o}$ removed). In each task, the stimulus duration was randomly perturbed across presentations (440±40 msec) to control for timing-based cues, and dots moved coherently through a radial speed

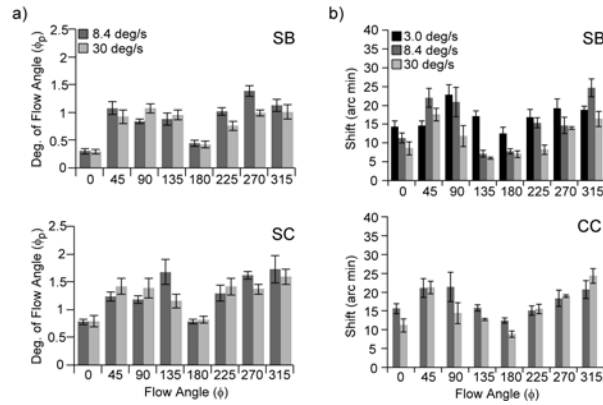

Figure 2: a) GMP thresholds across 8 'test' motions at two mean dot speeds for two observers. Performance varied continuously with thresholds for radial motions ($\phi$=0, 180°) significantly lower than those for circular motions ($\phi$=90,270°), (p<0.001; t(37)=3.39). b) COM thresholds at three mean dot speeds for two observers. As with the GMP task, performance varied continuously with thresholds for radial motions significantly lower than those for circular motions, (p<0.001; t(37)=4.47).

gradient in directions consistent with the global motion pattern presented. Discrimination thresholds were obtained across eight 'test' motions corresponding to expansion, contraction, CW and CCW rotation, and the four intermediate spiral motions. To minimize adaptation to specific motion patterns, opposing motions (e.g., expansion/ contraction) were interleaved across paired presentations.

## 2.1   Results

Discrimination thresholds are reported here from a subset of the observer population consisting of three experienced psychophysical observers, one of which was naïve to the purpose of the psychophysical tasks. For each condition, performance is reported as the mean and standard error averaged across 8-12 thresholds.

Across observers and dot speeds GMP thresholds followed a distinct trend in the stimulus space [13], with radial motions (expansion/contraction) significantly lower than circular motions (CW/CCW rotation), (p<0.001; t(37)=3.39), (Fig. 2a). While thresholds for the intermediate spiral motions were not significantly different from circular motions (p=0.223, t(60)=0.74), the trends across 'test' motions were well fit within the stimulus space (SB: r>0.82, SC: r>0.77) by sinusoids whose period and phase were 196 ± 10° and -72 ± 20° respectively (Fig. 1a).

When the radial speed gradient was removed by randomizing the spatial distribution of dot speeds, threshold performance increased significantly across observers (p<0.05; t(17)=1.91), particularly for circular motions (p<0.005; t(25)=3.31), (data not shown). Such performance suggests a perceptual contribution associated with the presence of the speed gradient and is particularly interesting given the fact that the speed gradient did not contribute computationally relevant information to the task. However, the speed gradient did convey information regarding the integrative structure of the global motion field and as such suggests a preference of the underlying motion mechanisms for spatially structured speed information.

Similar trends in performance were observed in the COM task across observers and dot speeds. Discrimination thresholds varied continuously as a function of the 'test'

motion with thresholds for radial motions significantly lower than those for circular motions, (p<0.001; t(37)=4.47) and could be well fit by a sinusoidal trend line (e.g. SB at 3 deg/s: r>0.91, period = $178 \pm 10^{o}$ and phase = $-70 \pm 25^{o}$), (Fig. 2b).

## 2.2   A local or global task?

The consistency of the cyclic threshold profile in stimuli that restricted the temporal integration of individual dot motions [13], and simultaneously contained all directions of motion, generally argues against a primary role for local motion mechanisms in the psychophysical tasks. While the psychophysical literature has reported a wide variety of "local" motion direction anisotropies whose properties are reminiscent of the results observed here, e.g. [14], all would predict equivalent thresholds for radial and circular motions for a set of uniformly distributed and/or spatially restricted motion direction mechanisms. Together with the computational impact of the speed gradient and psychophysical studies supporting the existence of wide-field motion pattern mechanisms [5, 6], these results suggest that the threshold differences across the GMP and COM tasks may be associated with variations in the computational properties across a series of specialized motion pattern mechanisms.

## 3   A computational model

The similarities between the motion pattern stimuli used to quantify human perception and the visual motion properties of cells in MSTd suggests that MSTd may play a computational role in the psychophysical tasks. To examine this hypothesis, we constructed a population of MSTd-like units whose visual motion properties were consistent with the reported neurophysiology (see [13] for details). Across the population, the distribution of receptive field centers was uniform across polar angle and followed a gamma distribution $\Gamma(5,6)$ across eccenticity [7]. For each unit, visual motion responses followed a gaussian tuning profile as a function of the stimulus flow angle $G(\phi)$, ($\sigma_i=60\pm30^{o}$; [10]), and the distance of the stimulus COM from the unit's receptive field center $G_{sat}(x_i, y_i, \sigma_s=19^{o})$, Eq. 1, such that its preferred motion response was position invariant to small shifts in the COM [10] and degraded continuously for large shifts [9].

Within the model, simulations were categorized according to the distribution of preferred motions represented across the population (one reported in MSTd and a uniform control). The first distribution simulated an expansion bias in which the density of preferred motions decreased symmetrically from expansions to contraction [10]. The second distribution simulated a uniform preference for all motions and was used as a control to quantify the effects of an expansion bias on psychophysical performance. Throughout the paper we refer to simulations containing these distributions as 'Expansion-biased' and 'Uniform' respectively.

### 3.1   Extracting perceptual estimates from the neural code

For each stimulus presentation, the $i^{th}$ unit's response was calculated as the average firing rate, $R_i$, from the product of its motion pattern and spatial tuning profiles,

$$R_i = R_{max}\ G\big(\min[\phi - \phi_i], \sigma_{t_i}\big)G_{sat_i}\big(x - x_i, y - y_i, \sigma_s\big) + P\big(\lambda = 12\big) \tag{1}$$

where $R_{max}$ is the maximum preferred stimulus response (spikes/s), min[ ] refers to the minimum angular distance between the stimulus flow angle $\phi$ and the unit's preferred motion $\phi_i$, $G_{sat}$ is the unit's spatial tuning profile saturated within the central $5\pm3^{o}$, $\sigma_{ti}$ and $\sigma_s$ are the standard deviations of the unit's motion pattern and

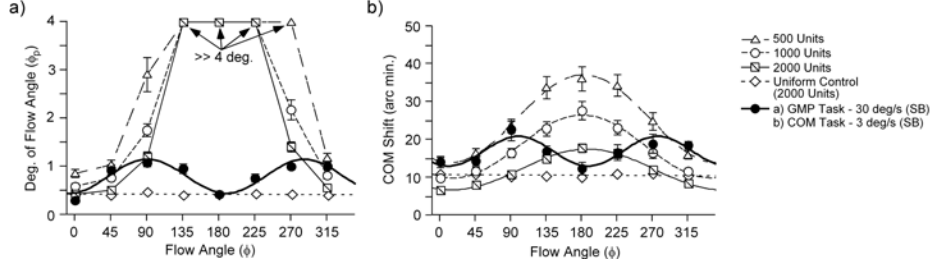

Figure 3: Model vs. psychophysical performance for independently responding units. Model thresholds are reported as the average (±1 S.E.) across five simulated populations. a) GMP thresholds were highest for contracting motions and lowest for expanding motions across all Expansion-biased populations. b) Comparable trends in performance were observed for COM thresholds. Comparison with the Uniform control simulations in both tasks (2000 units shown here) indicates that thresholds closely followed the distribution of preferred motions simulated within the model.

spatial tuning profiles respectively, $(x_i, y_i)$ is the spatial location of the unit's receptive field center, $(x,y)$ is the spatial location of the stimulus COM, and $P(\lambda=12)$ is the background activity simulated as an uncorrelated Poisson process.

The psychophysical tasks were simulated using a modified center-of-gravity approach to decode estimates of the stimulus properties, i.e. flow angle $(\hat{\phi})$ and COM location in the visual field $(\hat{x}, \hat{y})$, from the neural population

$$\left(\hat{x},\ \hat{y},\ \hat{\phi}\right) = \left( \frac{\sum_i x_i R_i}{\sum_i R_i},\ \frac{\sum_i y_i R_i}{\sum_i R_i},\ \sum_i \vec{\phi}_i R_i \right) \tag{2}$$

where $\vec{\phi}_i$ is the unit vector in the stimulus space (Fig. 1a) corresponding to the unit's preferred motion. For each set of paired stimuli, psychophysical judgments were made by comparing the estimated stimulus properties according to the discrimination criteria, specified in the psychophysical tasks. As with the psychophysical experiments, discrimination thresholds were computed using a least-squares fit to percent correct performance across constant stimulus levels.

## 3.2   Simulation 1: Independent neural responses

In the first series of simulations, GMP and COM thresholds were quantified across three populations (500, 1000, and 2000 units) of independently responding units for each simulated distribution (Expansion-biased and Uniform). Across simulations, both the range in thresholds and their trends across 'test' motions were compared with human psychophysical performance to quantify the effects of population size and an expansion biased preferred motion distribution on model performance.

Over the psychophysical range of interest ($\phi_p \pm 7^\circ$), GMP thresholds for contracting motions were at chance across all Expansion-biased populations, (Fig. 3a). While thresholds for expanding motions were generally consistent with those for human observers, those for circular motions remained significantly higher for all but the largest populations. Similar trends in performance were observed for the COM task, (Fig. 3b). Here the range of COM thresholds was well matched with human performance for simulations containing 1000 units, however, the trends across motion patterns remained inconsistent even for the largest populations.

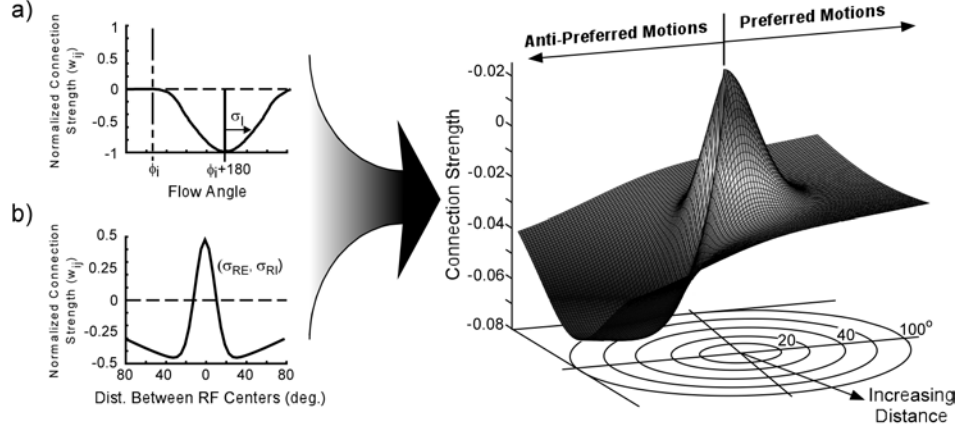

Figure 4: Proposed recurrent connection profile between motion pattern units. a) Across the motion pattern space connection strength followed an inverse gaussian profile such that the $i^{th}$ unit (with preferred motion $\phi_i$) systematically inhibited units with anti-preferred motions centered at $180+\phi_i$. b) Across the visual field connection strength followed a difference-of-gaussians profile as a function of the relative distance between receptive field centers such that spatially local units are mutually excitatory ($\sigma_{Re}=10^o$) and more distant units were mutually inhibitory ($\sigma_{Ri}=80^o$).

For simulations containing a uniform distribution of preferred motions, the threshold range was consistent with human performance on both tasks, however, the trend across motion patterns was generally flat. What variability did occur was due primarily to the discrete sampling of preferred motions across the population.

Comparison of the discrimination thresholds for the Expansion-biased and Uniform populations indicates that the trend across thresholds was closely matched to the underlying distributions of preferred motions. This result in due in part to the near-equal weighting of independently responding units and can be explained to a first approximation by the proportional increase in the signal-to-noise ratio across the population as a function of the density of units responsive to a given 'test' motion.

### 3.3 Simulation 2: An interconnected neural structure

In a second series of simulations, we examined the computational effect of adding recurrent connections between units. If the distribution of preferred motions in MSTd is in fact biased towards expansions, as the neurophysiology suggests, it seems unlikely that independent estimates of the visual motion information would be sufficient to yield the threshold profiles observed in the psychophysical tasks.

We hypothesize that a simple fixed architecture of excitatory and/or inhibitory connections is sufficient to account for the cyclic trends in discrimination thresholds. Specifically, we propose that a recurrent connection profile whose strength varies as a function of (a) the similarity between preferred motion patterns and (b) the distance between receptive field centers, is computationally sufficient to recover the trends in GMP/COM performance (Fig. 4),

$$w_{ij} = S_R \, e^{-\frac{(x_i-x_j)^2+(y_i-y_j)^2}{2\sigma_{Re}^2}} - \frac{S_R}{2} e^{-\frac{(x_i-x_j)^2+(y_i-y_j)^2}{2\sigma_{Ri}^2}} - S_\phi \, e^{\frac{-(min[\phi_i-\phi_j])^2}{2\sigma_I^2}} \qquad (3)$$

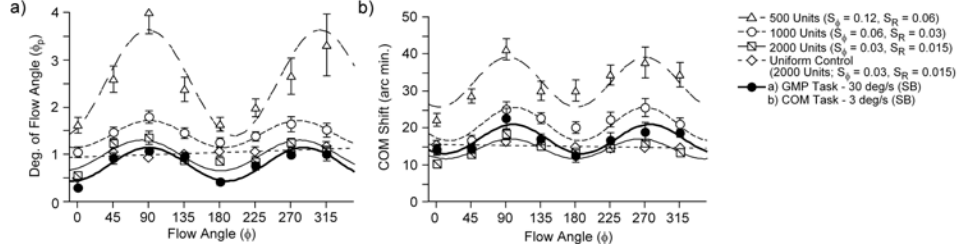

Figure 5: Model vs. psychophysical performance for populations containing recurrent connections ($\sigma_I=80^o$). As the number of units increased for Expansion-biased populations, discrimination thresholds decreased to psychophysical levels and the sinusoidal trend in thresholds emerged for both the (a) GMP and (b) COM tasks. Sinusoidal trends were established for as few as 1000 units and were well fit (r>0.9) by sinusoids whose periods and phases were ($193.8 \pm 11.7^o$, $-70.0 \pm 22.6^o$) and ($168.2 \pm 13.7^o$, $-118.8 \pm 31.8^o$) for the GMP and COM tasks respectively.

where $w_{ij}$ is the strength of the recurrent connection between $i^{th}$ and $j^{th}$ units, $(x_i,y_i)$ and $(x_j,y_j)$ denote the spatial locations of their receptive field centers, $\sigma_{Re}$ (=$10^o$) and $\sigma_{Ri}$ (=$80^o$) together define the spatial extent of a difference-of-gaussians interaction between receptive field centers, and $S_R$ and $S_\phi$ scale the connection strength. To examine the effects of the spread of motion pattern-specific inhibition and connection strength in the model, $\sigma_I$, $S_\phi$, and $S_R$ were considered free parameters.

Within the parameter space used to define recurrent connections (i.e., $\sigma_I$, $S_\phi$ and $S_R$), Monte Carlo simulations of Expansion-biased model performance (1000 units) yielded regions of high correlation on both tasks (with respect to the psychophysical thresholds, r>0.7) that were consistent across independently simulated populations. Typically these regions were well defined over a broad range such that there was significant overlap between tasks (e.g., for the GMP task ($S_R$=0.03), $\sigma_I$=[45,120$^o$], $S_\phi$=[0.03,0.3] and for the COM task ($\sigma_I$=80$^o$), $S_\phi$ = [0.03,0.08], $S_R$ = [0.005, 0.04]).

Fig. 5 shows averaged threshold performance for simulations of interconnected units drawn from the highly correlated regions of the ($\sigma_I$, $S_\phi$, $S_R$) parameter space. For populations not explicitly examined in the Monte Carlo simulations connection strengths ($S_\phi$, $S_R$) were scaled inversely with population size to maintain an equivalent level of recurrent activity. With the incorporation of recurrent connections, the sinusoidal trend in GMP and COM thresholds emerged for Expansion-biased populations as the number of units increased. In both tasks the cyclic threshold profiles were established for 1000 units and were well fit (r>0.9) by sinusoids whose periods and phases were consistent with human performance.

Unlike the Expansion-biased populations, Uniform populations were not significantly affected by the presence of recurrent connections (Fig. 5). Both the range in thresholds and the flat trend across motion patterns were well matched to those in Section 3.2. Together these results suggest that the sinusoidal trends in GMP and COM performance may be mediated by the combined contribution of the recurrent interconnections and the bias in preferred motions across the population.

## 4   Discussion

Using a biologically constrained computational model in conjunction with human psychophysical performance on two motion pattern tasks we have shown that the visual motion information encoded across an interconnected population of cells

responsive to motion patterns, such as those in MSTd, is computationally sufficient to extract perceptual estimates consistent with human performance. Specifically, we have shown that the cyclic trend in psychophysical performance observed across tasks, (a) cannot be reproduced using populations of independently responding units and (b) is dependent, in part, on the presence of an expanding motion bias in the distribution of preferred motions across the neural population.

The model's performance suggests the presence of specific recurrent structures within motion pattern responsive areas, such as MSTd, whose strength varies as a function of the similarity between preferred motion patterns and the distance between receptive field centers. While such structures have not been explicitly examined in MSTd and other higher visual motion areas there is anecdotal support for the presence of inhibitory connections [8]. Together, these results suggest that robust processing of the motion patterns associated with self-motion and optic flow may be mediated, in part, by recurrent structures in extrastriate visual motion areas whose distributions of preferred motions are biased strongly in favor of expanding motions.

## Acknowledgments

This work was supported by National Institutes of Health grant EY-2R01-07861-13 to L.M.V.

## References

[1]  Malach, R., Schirman, T., Harel, M., Tootell, R., & Malonek, D., (1997), *Cerebral Cortex*, 7(4): 386-393.

[2]  Gilbert, C. D., (1992), *Neuron*, 9: 1-13.

[3]  Koechlin, E., Anton, J., & Burnod, Y., (1999), *Biological Cybernetics*, 80: 25-44.

[4]  Stemmler, M., Usher, M., & Niebur, E., (1995), *Science*, 269: 1877-1880.

[5]  Burr, D. C., Morrone, M. C., & Vaina, L. M., (1998), *Vision Research*, 38(12): 1731-1743.

[6]  Meese, T. S. & Harris, S. J., (2002), *Vision Research*, 42: 1073-1080.

[7]  Tanaka, K. & Saito, H. A., (1989), *Journal of Neurophysiology*, 62(3): 626-641.

[8]  Duffy, C. J. & Wurtz, R. H., (1991), *Journal of Neurophysiology*, 65(6): 1346-1359.

[9]  Duffy, C. J. & Wurtz, R. H., (1995), *Journal of Neuroscience*, 15(7): 5192-5208.

[10] Graziano, M. S., Anderson, R. A., & Snowden, R., (1994), *Journal of Neuroscience*, 14(1): 54-67.

[11] Celebrini, S. & Newsome, W., (1994), *Journal of Neuroscience*, 14(7): 4109-4124.

[12] Celebrini, S. & Newsome, W. T., (1995), *Journal of Neurophysiology*, 73(2): 437-448.

[13] Beardsley, S. A. & Vaina, L. M., (2001), *Journal of Computational Neuroscience*, 10: 255-280.

[14] Matthews, N. & Qian, N., (1999), *Vision Research*, 39: 2205-2211.
